# A Connectionist Technique for Accelerated Textual Input: Letting a Network Do the Typing

**Dean A. Pomerleau**
pomerlea@cs.cmu.edu
School of Computer Science
Carnegie Mellon University
Pittsburgh, PA 15213

## Abstract

Each year people spend a huge amount of time typing. The text people type typically contains a tremendous amount of redundancy due to predictable word usage patterns and the text's structure. This paper describes a neural network system call AutoTypist that monitors a person's typing and predicts what will be entered next. AutoTypist displays the most likely subsequent word to the typist, who can accept it with a single keystroke, instead of typing it in its entirety. The multi-layer perceptron at the heart of AutoTypist adapts its predictions of likely subsequent text to the user's word usage pattern, and to the characteristics of the text currently being typed. Increases in typing speed of 2-3% when typing English prose and 10-20% when typing C code have been demonstrated using the system, suggesting a potential time savings of more than 20 hours per user per year. In addition to increasing typing speed, AutoTypist reduces the number of keystrokes a user must type by a similar amount (2-3% for English, 10-20% for computer programs). This keystroke savings has the potential to significantly reduce the frequency and severity of repeated stress injuries caused by typing, which are the most common injury suffered in today's office environment.

## 1 Introduction

People in general, and computer professionals in particular, spend a huge amount of time typing. Most of this typing is done sitting in front of a computer display using a keyboard as the primary input device. There are a number of efforts using artificial neural networks and other techniques to improve the comfort and efficiency of human-computer communication using alternative modalities. Speech recognition [Waibel et al., 1988], handwritten character recognition [LeCun et al., 1989], and even gaze tracking [Baluja & Pomerleau, 1993] have

the potential to facilitate this communication. But these technologies are still in their infancy, and at this point cannot approach the speed and accuracy of even a moderately skilled typist for textual input.

Is there some way to improve the efficiency of standard keyboard-based human-computer communication? The answer is yes, there are several ways to make typing more efficient. The first, called the Dvorak keyboard, has been around for over 60 years. The Dvorak keyboard has a different arrangement of keys, in which the most common letters, E, T, S, etc., are on the home row right under the typist's fingers. This improved layout requires the typist's fingers to travel 1/16th as far, resulting in an average of 20% increase in typing speed. Unfortunately, the de facto standard in keyboards is the inefficient QWERTY configuration, and people are reluctant to learn a new layout.

This paper describes another approach to improving typing efficiency, which can be used with either the QWERTY or DVORAK keyboards. It takes advantage of the hundreds of thousands of computer cycles between the typist's keystrokes which are typically wasted while the computer idly waits for additional input. By spending those cycles trying to predict what the user will type next, and allowing the typist to accept the prediction with a single keystroke, substantial time and effort can be saved over typing the entire text manually.

There are actually several such systems available today, including a package called "Autocompletion" developed for gnu-emacs by the author, and an application called "Magic Typist" developed for the Apple Macintosh by Olduvai Software. Each of these maintains a database of previously typed words, and suggests completions for the word the user is currently in the middle of typing, which can be accepted with a single keystroke. While reasonable useful, both have substantial drawbacks. These systems use a very naive technique for calculating the best completion, simply the one that was typed most recently. In fact, experiments conducted for this paper indicated that this "most recently used" heuristic is correct only about 40% of the time. In addition, these two systems are annoyingly verbose, always suggesting a completion if a word has been typed previously which matches the prefix typed so far. They interrupt the user's typing to suggest a completion even if the word they suggest hasn't been typed in many days, and there are many other alternative completions for the prefix, making it unlikely that the suggestion will be correct. These drawbacks are so severe that these systems frequently decrease the user's typing speed, rather than increase it.

The AutoTypist system described in this paper employs an artificial neural network during the spare cycles between keystrokes to make more intelligent decisions about which completions to display, and when to display them.

## 2   The Prediction Task

To operationalize the goal of making more intelligent decisions about which completions to display, we have defined the neural networks task to be the following: Given a list of candidate completions for the word currently being typed, estimate the likelihood that the user is actually typing each of them. For example, if the user has already types the prefix "aut", the word he is trying to typing could any one of a large number of possibilities, including "autonomous", "automatic", "automobile" etc. Given a list of these possibilities taken from a dictionary, the neural network's task is to estimate the probability that each of these is the word the user will type.

A neural network cannot be expected to accurately estimate the probability for a particular completion based on a unique representation for each word, since there are so many words

| ATTRIBUTE | DESCRIPTION |
|---|---|
| absolute age | time since word was last typed |
| relative age | ratio of the words age to age of the most recently typed alternative |
| absolute frequency | number of times word has been typed in the past |
| relative frequency | ratio of the words frequency to that of the most often typed alternative |
| typed previous | 1 if user has typed word previously, 0 otherwise |
| total length | the word's length, in characters |
| remaining length | the number of characters left after the prefix to be typed for this word |
| special character match | the percentage of "special characters" (i.e. not a-z) in this word relative to the percentage of special characters typed recently |
| capitalization match | 1 if the capitalization of the prefix the user has already typed matches the word's usual capitalization, 0 otherwise. |

Table 1: Word attributes used as input to the neural network for predicting word probabilities.

in the English language, and there is only very sparse data available to characterize an individual's usage pattern for any single word. Instead, we have chosen to use an input representation that contains only those characteristics of a word that could conceivably have an impact on its probability of being typed. The attributes we employed to characterize each completion are listed in Table 1.

These are not the only possible attributes that could be used to estimate the probability of the user typing a particular word. An additional characteristic that could be helpful is the word's part of speech (i.e. noun, verb, adjective, etc.). However this attribute is not typically available or even meaningful in many typing situations, for instance when typing computer programs. Also, to effectively exploit information regarding a word's part of speech would require the network to have knowledge about the context of the current text. In effect, it would require at least an approximate parse tree of the current sentence. While there are techniques, including connectionist methods [Jain, 1991], for generating parse trees, they are prone to errors and computationally expensive. Since word probability predictions in our system must occur many times between each key the user types, we have chosen to utilize only the easy to compute attributes shown in Table 1 to characterize each completion.

## 3  Network Processing

The network architecture employed for this system is a feedforward multi-layer perceptron. Each of the networks investigated has nine input units, one for each of the attributes listed in Table 1, and a single output unit. As the user is typing a word, the prefix he has typed so far is used to find candidate completions from a dictionary, which contains 20,000 English words plus all words the user has typed previously. For each of these candidate completions, the nine attributes in Table 1 are calculated, and scaled to the range of 0.0 to 1.0. These values become the activations of the nine units in the input layer. Activation is propagated through the network to produce an activation for the single output unit, representing the

probability that this particular candidate completion is the one the user is actually typing. These candidate probabilities are then used to determine which (if any) of the candidates should be displayed to the typist, using a technique described in a later section.

To train the network, the user's typing is again monitored. After the user finishes typing a word, for each prefix of the word a list of candidate completions, and their corresponding attributes, is calculated. These form the input training patterns. The target activation for the single output unit on a pattern is set to 1.0 if the candidate completion represented by that pattern is the word the user was actually typing, and 0.0 if the candidate is incorrect. Note that the target output activation is binary. As will be seen below, the actual output the network learns to produce is an accurate estimate of the completion's probability. Currently, training of the network is conducted off-line, using a fixed training set collected while a user types normally. Training is performed using the standard backpropagation learning algorithm.

## 4   Experiments

Several tests were conducted to determine the ability of multi-layer perceptrons to perform the mapping from completion attributes to completion probability. In each of the tests, networks were trained on a set of input/output exemplars collected over one week of a single subject's typing. During the training data collection phase, the subject's primary text editing activities involved writing technical papers and composing email, so the training patterns represent the word choice and frequency distributions associated with these activities. This training set contained of 14,302 patterns of the form described above.

The first experiment was designed to determine the most appropriate network architecture for the prediction task. Four architecture were trained on a 10,000 pattern subset of the training data, and the remaining 4,302 patterns were used for cross validation. The first of the four architectures was a perceptron, with the input units connected directly to the single output unit. The remaining three architectures had a single hidden layer, with three, six or twelve hidden units. The networks with hidden units were fully connected without skip connections from inputs to output. Networks of three and six hidden units which included skip connections were tested, but did not exhibit improved performance over the networks without skip connections, so they are not reported.

Each of the network architectures were trained four times, with different initial random weights. The results reported are those produced by the best set of weights from these trials. Note that the variations between trials with a single architecture were small relative to the variations between architectures. The trained networks were tested on a disjoint set of 10,040 collected while the same subject was typing another technical paper.

Three different performance metrics were employed to evaluate the performance of these architectures on the test set. The first was the standard mean squared error (MSE) metric, depicted in Figure 1. The MSE results indicate that the architectures with six and twelve hidden units were better able to learn the task than either the perceptron, or the network with only three hidden units. However the difference appears to be relatively small, on the order of about 10%.

MSE is not a very informative error metric, since the target output is binary (1 if the completion is the one the user was typing, 0 otherwise), but the real goal is to predict the probability that the completion is correct. A more useful measure of performance is shown in Figure 2. For each of the four architectures, it depicts the predicted probability that a completion is correct, as measured by the network's output activation value, vs. the

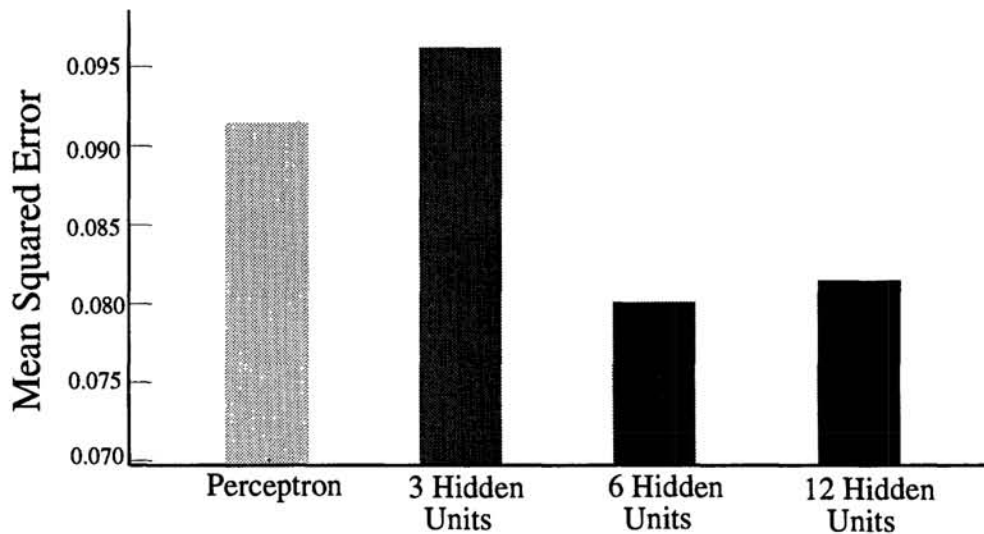

Figure 1: Mean squared error for four networks on the task of predicting completion probability.

actual probability that a completion is correct. The lines for each of the four networks were generated in the following manner. The network's output response on each of the 10,040 test patterns was used to group the test patterns into 10 categories. All the patterns which represented completions that the network predicted to have a probability of between 0 and 10% of being correct (output activations of 0.0-0.1) were placed in one category. Completions that the network predicted to have a 10-20% change of being right were placed in the second category, etc. For each of these 10 categories, the actual likelihood that a completion classified within the category is correct was calculated by determining the percent of the completions within that category that were actually correct.

As a concrete example, the network with 6 hidden units produced an output activation between 0.2 and 0.3 on 861 of the 10,040 test patterns, indicating that on these patterns it considered there to be a 20-30% chance that the completion each pattern represented was the word the user was typing. On 209 of these 861 patterns in this category, the completion was actually the one the user was typing, for a probability of 24.2%. Ideally, the actual probability should be 25%, half way between the minimum and maximum predicted probability thresholds for this category. This ideal classification performance is depicted as the solid 45° line labeled "Target" in Figure 2. The closer the line for a given network matches this 45° line, the more the network's predicted probability matches the actual probability for a completion. Again, the networks with six and twelve hidden units outperformed the networks with zero and three hidden units, as illustrated by their much smaller deviations from the 45° line in Figure 2.

The output activations produced by the networks with six and twelve hidden units reflect the actual probability that the completion is correct quite accurately. However prediction accuracy is only half of what is required to perform the final system goal, which recall was to identify as many high probability completions as possible, so they can be suggested to the user without requiring him to manually type them. If overall accuracy of the probability predictions were the only requirement, a network could score quite highly by classifying

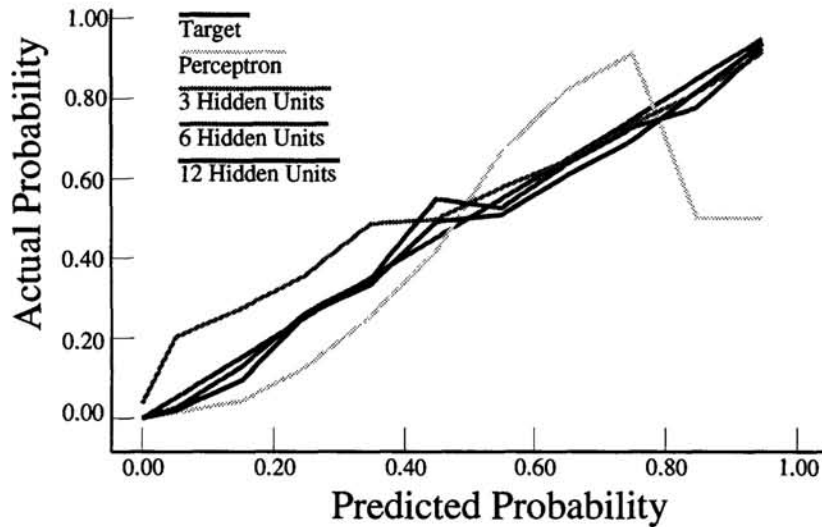

Figure 2: Predicted vs. actual probability of a completion being correct for the four architectures tested.

every pattern into the 10-20% category, since about 15% of the 10,040 completions in the test set represent the word the user was typing at the time. But a constant prediction of 10-20% probability on every alternative completion would not allow the system to identify and suggest to the user those individual completions that are much more likely than the other alternatives.

To achieve the overall system goal, the network must be able to accurately identify as many high probability completions as possible. The ability of each of the four networks to achieve this goal is shown in Figure 3. This figures shows the percent of the 10,040 test patterns each of the four networks classified as having more than a 60% probability of being correct. The 60% probability threshold was selected because it represents a level of support for a single completion that is significantly higher than the support for all the others. As can be seen in Figure 3, the networks with hidden units again significantly outperformed the perceptron, which was able to correctly identify fewer than half as many completions as highly likely.

## 5   AutoTypist System Architecture and Performance

The networks with six and twelve hidden units are able to accurately identify individual completions that have a high probability of being the word the user is typing. In order to exploit this prediction ability and speed up typing, we have build an X-window based application called AutoTypist around the smaller of the two networks. The application serves as the front end for the network, monitoring the user's typing and identifying likely completions for the current word between each keystroke. If the network at the core of AutoTypist identifies a single completion that it is both significantly more probably than all the rest, and also longer than a couple characters, it will momentarily display the completion after the current cursor location in whatever application the user is currently typing[1]. If the displayed completion is the word the user is typing, he can accept it with a single keystroke

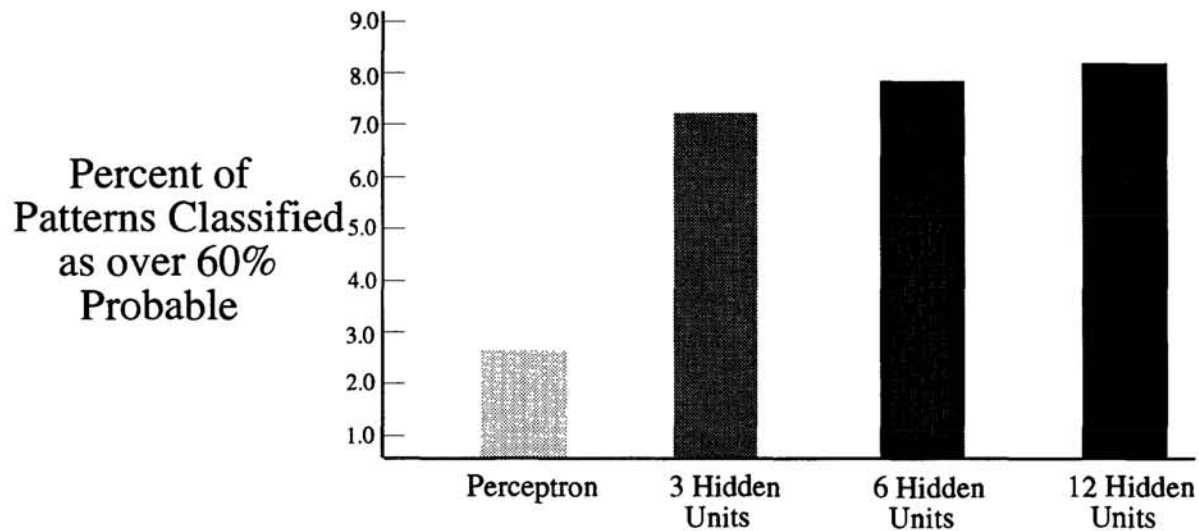

Figure 3: Percent of candidate completions classified as having more than a 60% chance of being correct for the four architectures tested.

and move on to typing the next word. If the displayed completion is incorrect, he can continue typing and the completion will disappear.

Quantitative results with the fully integrated AutoTypist system, while still preliminary, are very encouraging. In a two week trial with two subjects, who could type at 40 and 60 wpm without AutoTypists, their typings speeds were improved by 2.37% and 2.21% respectively when typing English text. Accuracy improvements during these trials were even larger, since spelling mistakes become rare when AutoTypist is doing a significant part of the typing automatically. When writing computer programs, speed improvements of 12.93% and 18.47% were achieved by the two test subjects. This larger speedup was due to the frequent repetition of variable and function names in computer programs, which AutoTypist was able to expedite. Not only is computer code faster to produce with AutoTypist, it is also easier to understand. AutoTypist encourages the programmer to use long, descriptive variable and function names, by making him type them in their entirety only once. On subsequent instances of the same name, the user need only type the first few characters and then exploit AutoTypist's completion mechanism to type the rest. These speed improvements were achieved by subjects who are already relatively proficient typists. Larger gains can be expected for less skilled typists, since typing an entire word with a single keystroke will save more time when each keystroke takes longer.

Perhaps an even more significant benefit results from the reduced number of keystrokes AutoTypist requires the user to type. During the test trials described above, the two test subjects had to strike an average of 2.89% fewer keys on the English text, and 16.42% fewer keys on the computer code than would have been required to type the text out in its entirety. Clearly this keystroke savings has the potential to benefit typists who suffer from repeated stress injuries brought on by typing.

Unfortunately it is impossible to quantitatively compare these results with those of the other completion-based typing aids described in the introduction, since the other systems have not been quantitatively evaluated. Subjectively, AutoTypist is far less disturbing than the

alternatives, since it only displays a completion when there is a very good chance it is the correct one.

## 6  Future Work

Further experiments are required to verify the typing speed improvements possible with AutoTypist, and to compare it with alternative typing improvement systems. Preliminary experiments suggest a network trained on the word usage patterns of one user can generalize to that of other users, but it may be necessary to train a new network for each individual typist. Also, the experiments conducted for this paper indicate that a network trained on one type of text, English prose, can generalize to text with quite different word frequency patterns, C language computer programs. However substantial prediction improvements, and therefore typing speedup, may be possible by training separate networks for different types of text. The question of how to rapidly adapt a single network, or perhaps a mixture of expert networks, to new text types is one which should be investigated.

Even without these extensions, AutoTypist has the potential to greatly improve the comfort and efficiency of the typing tasks. For people who type English text two hours per workday, even the conservative estimate of a 2% speedup translates into 10 hours of savings per year. The potential time savings for computer programming is even more dramatic. A programmer who types code two hours per workday could potentially save between 52 and 104 hours in a single year by using AutoTypist. With such large potential benefits, commercial development of the AutoTypist system is also being investigated.

### Acknowledgements

I would like to thank David Simon and Martial Hebert for their helpful suggestions, and for acting as willing test subjects during the development of this system.

## Footnotes

[1]The criterion for displaying a completion, and the human interface for AutoTypist, are somewhat more sophisticated than this description. However for the purposes of this paper, a high level description is sufficient.

## References

[Baluja & Pomerleau, 1993] Baluja, S. and Pomerleau, D.A. (1993) Non-Intrusive Gaze Tracking Using Artificial Neural Networks. In *Advances in Neural Information Processing Systems 6*, San Mateo, CA: Morgan Kaufmann Publishers.

[Jain, 1991] Jain, A.N. (1991) PARSEC: A connectionist learning architecture for parsing spoken language. Carnegie Mellon University School of Computer Science Technical Report CMU-CS-91-208.

[LeCun et al., 1989] LeCun, Y., Boser, B., Denker, J.S., Henderson, D., Howard, R.E., Hubbard, W., and Jackel, L.D. (1989) Backpropagation applied to handwritten zip code recognition. *Neural Computation 1(4)*.

[Waibel et al., 1988] Waibel, A., Hanazawa, T., Hinton, G., Shikano, K., Lang, K. (1988) Phoneme recognition: Neural Networks vs. Hidden Markov Models. *Proceedings from Int. Conf. on Acoustics, Speech and Signal Processing*, New York, New York.
